# Attentional Processing on a Spike-Based VLSI Neural Network

**Yingxue Wang, Rodney Douglas, and Shih-Chii Liu**
Institute of Neuroinformatics
University of Zurich and ETH Zurich
Winterthurerstrasse 190
CH-8057 Zurich, Switzerland
yingxue,rjd,shih@ini.phys.ethz.ch

## Abstract

The neurons of the neocortex communicate by asynchronous events called action potentials (or 'spikes'). However, for simplicity of simulation, most models of processing by cortical neural networks have assumed that the activations of their neurons can be approximated by event rates rather than taking account of individual spikes. The obstacle to exploring the more detailed spike processing of these networks has been reduced considerably in recent years by the development of hybrid analog-digital Very-Large Scale Integrated (hVLSI) neural networks composed of spiking neurons that are able to operate in real-time. In this paper we describe such a hVLSI neural network that performs an interesting task of selective attentional processing that was previously described for a simulated 'pointer-map' rate model by Hahnloser and colleagues. We found that most of the computational features of their rate model can be reproduced in the spiking implementation; but, that spike-based processing requires a modification of the original network architecture in order to memorize a previously attended target.

## 1 Introduction

The network models described in the neuroscience literature have frequently used rate equations to avoid the difficulties of formulating mathematical descriptions of spiking behaviors; and also to avoid the excessive computational resources required for simulating spiking networks. Now, the construction of multi-chip hybrid VLSI (hVLSI) systems that implement large-scale networks of real-time spiking neurons and spike-based sensors is rapidly becoming a reality [3–5, 7], and so it becomes possible to explore the performance of event-based systems in various processing tasks and network behavior of populations of spiking, rather than rate neurons.

In this paper we use an hVLSI network to implement a spiking version of the 'pointer-map' architecture previously described for rate networks by Hahnloser and colleagues [2]. In this architecture, a small number of pointer neurons are incorporated in the feedback of a recurrently connected network. The pointers steer the feedback onto the map, and so focus processing on the attended map neurons. This is an interesting architecture because it reflects a general computational property of sensorimotor and attentional/intentional processing based on pointing. Directing attention, foveating eyes, and reaching limbs all appeal to a pointer like interaction with the world, and such pointing is known to modulate the responses of neurons in a number of cortical and subcortical areas.

The operation of the pointer-map depends on the steady focusing of feedback on the map neurons during the period of attention. It is easy to see how this steady control can be achieved in the neurons have continuous, rate outputs; but it is not obvious whether this behavior can be achieved also with intermittently spiking neural outputs. Our objective was thus to evaluate whether networks

of spiking neurons would be able to combine the benefits of both event-based processing and the attentional properties of pointer-map architecture.

## 2  Pointer-Map Architecture

A pointer-map network consists of two reciprocally connected populations of excitatory neurons. Firstly, there is a large population of map neurons that for example provide a place encoding of some variable such as the orientation of a visual bar stimulus. A second, small population of pointer neurons exercises attentional control on the map. In addition to the reciprocal connections between the two populations, the map neurons receive feedforward (e.g. sensory) input; and the pointer neurons receive top-down attentional inputs that instruct the pointers to modulate the location and intensity of the processing on the map (see Fig. 1(a)). The important functional difference between conventional recurrent networks (equivalently, 'recurrent maps') and the pointer-map, is that the pointer neurons are inserted in the feedback loop, and so are able to modulate the effect of the feedback by their top-down inputs. The usual recurrent excitatory connections between neurons are replaced in the pointer-map by recurrent connections between the map neurons and the pointer neurons that have sine and cosine weight profiles. Consequently, the activities of the pointer neurons generate a vectorial pattern of recurrent excitation whose direction points to a particular location on the map (Fig. 1(b)). Global inhibition provides competition between the map neurons, so that overall the pointer-map behaves as an attentionally selective soft winner-take-all network.

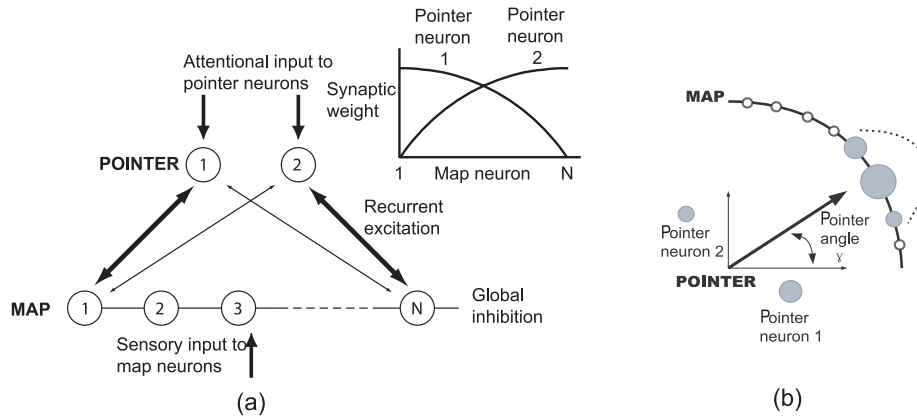

Figure 1: Pointer-map architecture. (a) Network consists of two layers of excitatory neurons. The map layer receives feedforward sensory inputs and inputs from two pointer neurons. The pointer neurons receive top-down attentional inputs and also inputs from the map layer. The recurrent connections between the map neurons and pointer neurons are set according to sine and cosine profiles. (b) The interaction between pointer neurons and map neurons. Each circle indicates the activity of one neuron. Clear circles indicate silent neurons and the sizes of the gray circles are proportional to the activation of the active neurons. The vector formed by the activities of the two pointer neurons on this angular plot points in the direction (the pointer angle $\gamma$) of the map neurons where the pointer-to-map input is the largest. The map-to-pointer input is proportional to the population vector of activities of the map neurons.

## 3  Spiking Network Chip Architecture

We implemented the pointer-map architecture on a multi-neuron transceiver chip fabricated in a 4-metal, 2-poly $0.35\mu$m CMOS process. The chip (Fig. 2) has 16 VLSI integrate-and-fire neurons, of which one acts as the global inhibitory neuron. Each neuron has 8 input synapses (excitatory and inhibitory). The circuit details of the soma and synapses are described elsewhere [4].

Input and output spikes are communicated within and between chips using the asynchronous Address Event Representation (AER) protocol [3]. In this protocol the action potentials that travel along point-to-point axonal connections are replaced by digital addresses on a bus that are usually

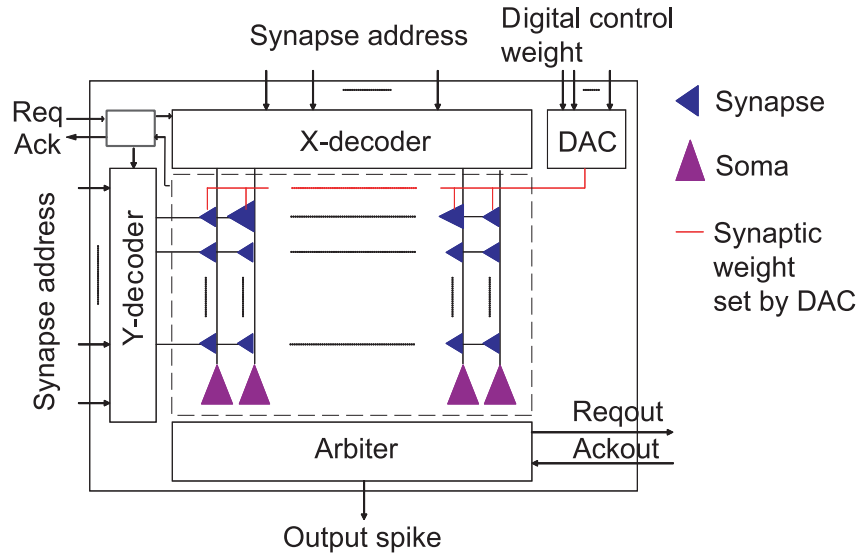

Figure 2: Architecture of multi-neuron chip. The chip has 15 integrate-and-fire excitatory neurons and one global inhibitory neuron. Each neuron has 8 input synapses. Input spikes and output spikes are communicated using an asynchronous handshaking protocol called Address Event Representation. When an input spike is to be sent to the chip, the handshaking signals, Req and Ack, are used to ensure that only valid addresses on a common digital bus are latched and decoded by X- and Y-decoders. The arbiter block arbitrates between all outgoing neuron spikes; and the neuron spike is sent off as the address of the neuron on a common digital bus through two handshaking signals (Reqout and Ackout). The synaptic weights of 2 out of the 8 synapses can be specified uniquely through an on-chip Digital-to-Analog converter that sets the synaptic weight of each synapse before that particular synapse is stimulated. The synaptic weight is specified as part of the digital address that normally codes the synaptic address.

the labels of source neurons and/or target synapses. In our chip, five bits of the AER address space are used to encode also the synaptic weight [1]. An on-chip Digital-to-Analog Converter (DAC) transforms the digital weights into the analog signals that set the individual efficacy of the excitatory synapses and inhibitory synapses for each neuron (Fig. 2).

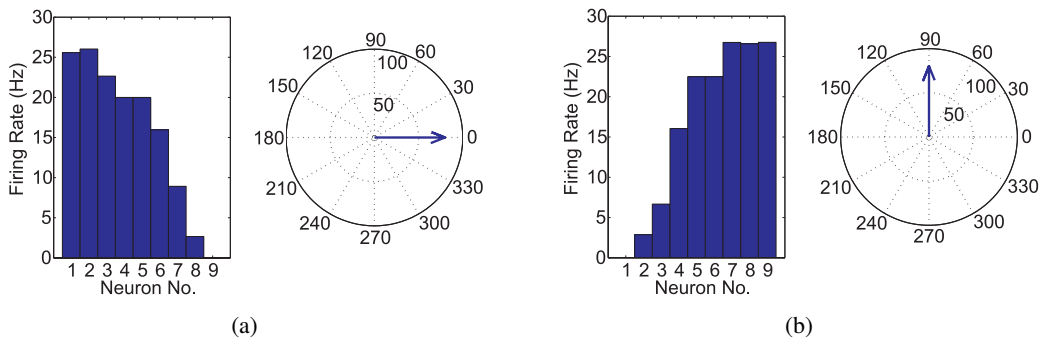

Figure 3: Resulting spatial distribution of activity in map neurons in response to attentional input to pointer neurons. The frequencies of the attentional inputs to P1, P2 are (a) [200Hz, 0Hz] (b) [0Hz,200Hz]. The y-axis shows the firing rate (Hz) of the map neurons (1–9) listed on the x-axis. The polar plot on the side of each figure shows the pointer angle $\gamma$ described by the pointer neuron activities.

# 4   Experiments

Our pointer-map was composed of a total of 12 neurons: 2 served as pointer neurons; 9 as map neurons; and 1 as the global inhibitory neuron. The synaptic weights of these neurons have a coefficient of variance in synaptic efficacy of about 0.25 due to silicon process variations. Through the on-chip DAC, we were able to reduce this variance for the excitatory synapses by a factor of 10. We did not compensate for the variance in the inhibitory synapses because it was technically more challenging to do that. The synaptic weights from each pointer neuron to every map neuron $j = 1, 2, ..., 9$ (Fig. 4(a)) were set according to the profile shown in Fig. 1(a). We compared the match between the programmed spatial connnectivity and the desired sine/cosine profile by activating only the top-down connections from the pointer neurons to the map neurons, while the bottom-up connections from the map neurons to the pointer neurons, and global inhibition were inactivated. In fact, because of the lower signal resolution, chip mismatch, and system noise, the measured profiles were only a qualitative match to a sine and cosine (Fig. 3), and the worst case variation from the ideal value was up to 50% for very small weights. Nevertheless, in spite of the imperfect match, we were able to reproduce most of the observations of [2].

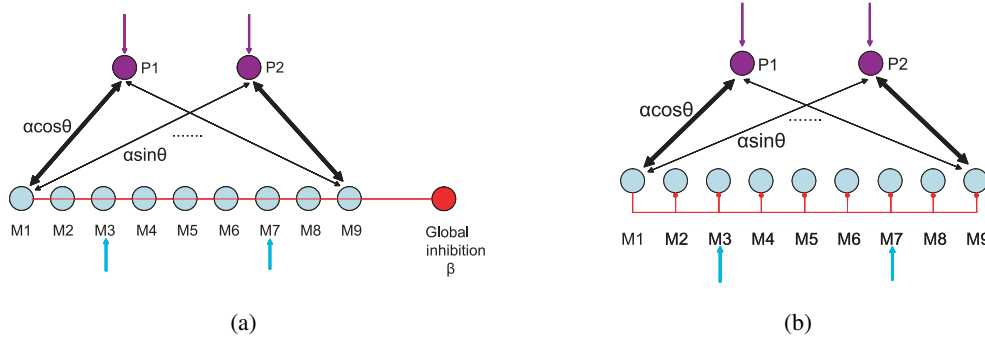

Figure 4: Network architecture used on the chip for attentional modulation. (a) Originally proposed pointer-map architecture. (b) New network architecture with no requirement for strong excitatory recurrent connections. The global inhibition is now replaced by neuron-specific inhibition.

## 4.1   Attentional Input Control

We tested the attentional control of the pointer neurons for this network (Fig. 4(a)) with activated recurrent connections and global inhibition. In addition, a common small constant input was applied to all map neurons. The location and activity on the map layer can be steered via the inputs to the pointer neurons, as seen in the three examples of Fig. 5. These results are similar to those observed by Hahnloser and colleagues in their rate model.

## 4.2   Attentional Target Selection

One computational feature of the rate pointer-map is its multistablity: If two or more sensory stimuli are presented to the network, strong attentional inputs to the pointer can select one of these stimuli even if the stimulus is not the strongest one. The preferred stimulus depends on the initial activities of the map and pointer neurons. Moreover, attentional inputs can steer the attention to a different location on the map, even after a stronger stimulus is selected initially. We repeated these experiments (Fig. 4 of [2]) for the spiking network. In our experiments, only two map neurons received feed-forward sensory inputs which consist of two regular spike trains of different frequencies. As shown in Fig. 6(a), the map neuron with the stronger feedforward input was selected. Attention could be steered to a different part of the map array by providing the necessary attentional inputs. And, the map neuron receiving the weaker stimulus could suppress the activity of another map neuron.

Furthermore, the original rate model can produce attentional memorization effects, that is, the location of the map layer activity is retained even after the inputs to the pointer neurons are withdrawn.

However, we were unsuccessful in duplicating the results of these experiments (see Fig. 6(a)) because the recurrent connection strength parameter $\alpha$ had to be greater than 1.

To explain why this strong recurrent connection strength was necessary, we first describe the steady state rate activities $M_1$ and $M_2$, of two arbitrary map neurons that are active:

$$M_1 = \lfloor m_1 - \beta (M_1 + M_2) + \alpha (\cos\theta_1 P_1 + \sin\theta_1 P_2)\rfloor_+ \tag{1}$$

$$M_2 = \lfloor m_2 - \beta (M_1 + M_2) + \alpha (\cos\theta_2 P_1 + \sin\theta_2 P_2)\rfloor_+ \tag{2}$$

where $P_1$, $P_2$ are the steady state rate activities of the pointer neurons; $m_1, m_2$ are the activities of the map neurons due to the sensory inputs; $\beta$ and $\alpha$ determine the strength of inhibitory and excitatory connections respectively; $\alpha\cos\theta_i$ and $\alpha\sin\theta_i$ determine the connection strengths between pointer neurons and map neurons $i$ for $\theta_i \in [0^o, 90^o]$.

The activities of the pointer neurons are given by

$$P_1 = \lfloor p_1 + \alpha (\cos\theta_1 M_1 + \cos\theta_2 M_2)\rfloor_+ \tag{3}$$

$$P_2 = \lfloor p_2 + \alpha (\sin\theta_1 M_1 + \sin\theta_2 M_2)\rfloor_+ \tag{4}$$

where $p_1$ and $p_2$ are the acitivities induced by inputs to the two pointer neurons.

Through substitution of Eqn.(3)(4) into Eqn.(1)(2) respectively, and assuming $p_1, p_2 = 0$, it shows that in order to satisfy the condition that $M_1 > M_2$ for $m_1 < m_2$, we need

$$\alpha > \frac{1}{\sqrt{1 - \cos(\theta_1 - \theta_2)}} > 1. \tag{5}$$

There are several factors that make it difficult for us to reproduce the attentional memorization experiments. Firstly, since we are only using a small number of neurons, each input spike has to create more than one output spike from a neuron in order to satisfy the above condition. On the one hand, this is very hard to implement, because the neurons have a refractory period, any input currents during this time will not influence the neuron. It means that we can not use self-excitation to get an effective $\alpha > 1$. On the other hand, even for $\alpha = 1$ (one input spike causes one output spike), it can easily lead to instability in the network because the timing of the arrival of the inhibitory and the excitatory inputs becomes a critical factor of the system stability. Secondly, the network has to operate in a hard winner-take-all mode because of the variance in the inhibitory synaptic efficacies. This means that the neuron is reset to its resting potential whenever it receives an inhibitory spike, thus removing all memory.

### 4.3 Attentional Memorization

By modifying the network architecture (see Fig. 4(b)), we were able to avoid using the strong excitatory connections as required in the original network. In our modified architecture, the inhibition is no longer global. Instead, each neuron inhibits all other neurons in the map population but itself. The steady state rate activities $M_1$ and $M_2$ are now given by

$$M_1 = \lfloor m_1 - \beta M_2 + \alpha (\cos\theta_1 P_1 + \sin\theta_1 P_2)\rfloor_+ \tag{6}$$

$$M_2 = \lfloor m_2 - \beta M_1 + \alpha (\cos\theta_2 P_1 + \sin\theta_2 P_2)\rfloor_+ \tag{7}$$

The equations for the steady-state pointer neuron activities $P_1$ and $P_2$ remain as before. The new condition for $\alpha$ is now

$$\alpha > \frac{1 - \beta}{\sqrt{1 - \cos(\theta_1 - \theta_2)}} \tag{8}$$

which means that $\alpha$ can be smaller than one. The intuitive explanation for the decrease of $\alpha$ is that, in the original architecture, the global inhibition inhibits all the map neurons including the winner. Therefore, in order to memorize the attended stimulus, the excitatory connections need to be strengthen to compensate for the self inhibition. But in the new scenario, we delete the self inhibitions, which then releases the requirement for strong excitations.

Using this new architecture, we performed the same experiments as described in Section 4.2 and we were now able to demonstrate attentional memorization. That is, the attended neuron with the weaker sensory input stimulus survived even after the attentional inputs were withdrawn. The same qualitative results were obtained even if all the remaining map neurons had a low background firing rate which mimic the effect of weak sensory inputs to different locations.

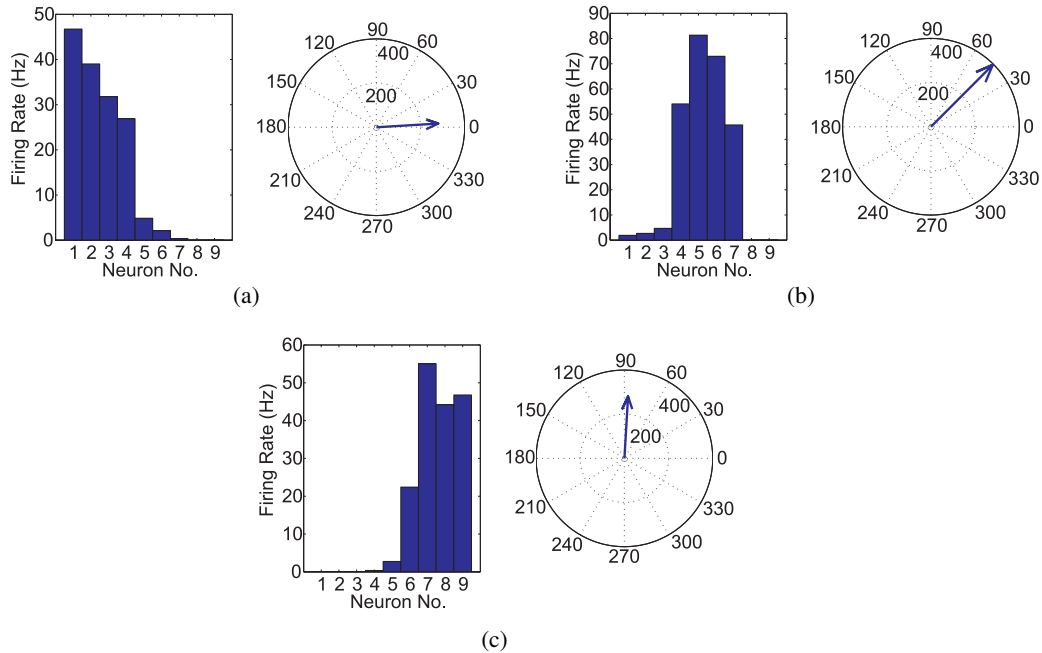

Figure 5: Results of experiments showing responses of map neurons for 3 settings of input strengths to the pointer neurons. Each map neuron has a background firing rate of 30Hz measured in the absence of activated recurrent connections and global inhibition. The attentional inputs to pointer neurons P1 and P2 are (a) [700Hz,50Hz], (b) [700Hz,700Hz], (c) [50Hz,700Hz]. The y-axis shows the firing rate (Hz) of the map neurons (1–9) listed on the x-axis.

# 5  Conclusion

In this paper, we have described a hardware 'pointer-map' neural network composed of spiking neurons that performs an interesting task of selective attentional processing previously described in a simulated 'pointer-map' rate model by Hahnloser and colleagues.

Neural network behaviors in computer simulations that use rate equations would likely be observed also in spiking networks if many input spikes can be integrated before the post-synaptic neuron's threshold is reached. However, extensive integration is not possible for practical electronic networks, in which there are relatively small numbers of neurons and synapses. We were find that most of the computational features of their simulated rate model could be reproduced in our hardware spiking implementation despite imprecisions of synaptic weights, and the inevitable fabrication related variability in the performance of individual neurons.

One significant difference between our spiking implementation and the rate model is the mechanism required to memorize a previously attended target. In our spike-based implementation, it was necessary to modify the original pointer-map architecture so that the inhibition no longer depends on a single global inhibitory neuron. Instead, each excitatory neuron inhibits all other neurons in the map population but itself.

Unfortunately, this approximate equivalence between excitatory and inhbitory neurons is inconsistent with the anatomical observation that only about 15% of cortical neurons are inhibitory. However, the original architecture could probably work if we had larger populations of map neurons, more synapses, and/or NMDA-like synapses with longer time constants. This is a scenario that we will explore in the future along with better characterization of the switching time dynamics of the attentional memorization experiments.

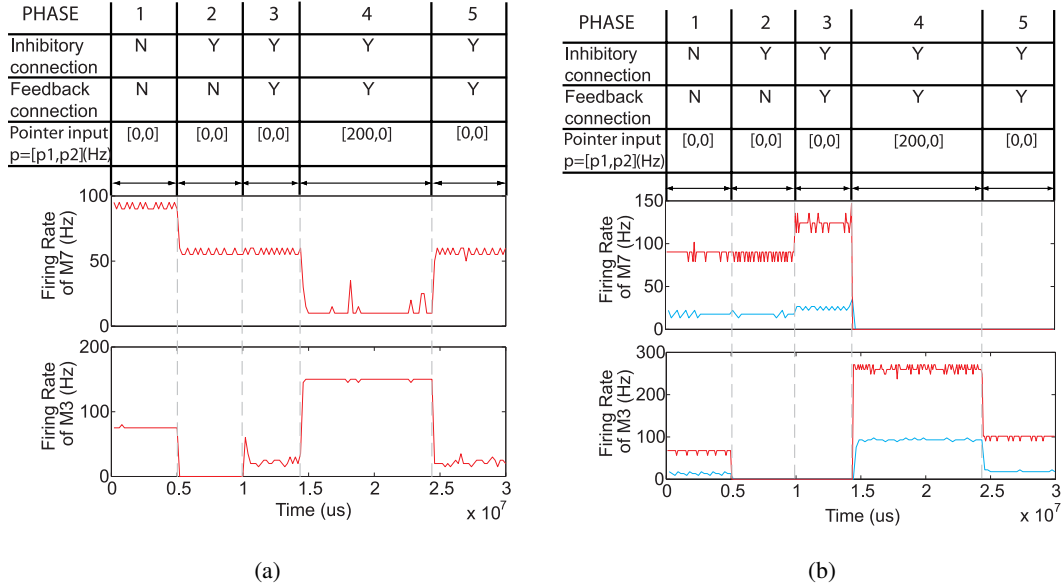

|  | (a) |  |  |  | (b) |

Figure 6: Results of attentional memorization experiments using the two different architectures in Fig. 4. (a) Results from original architecture. The sensory inputs to two map neurons M3 and M7 were set to [200Hz,230Hz]. The experiment was divided into 5 phases. In phase 1, the bottom-up connections and inhibitory connections were inactivated. In phase 2, the inhibitory connections were activated thus map neuron M3 which received the weaker input, was suppressed. In phase 3, the bottom-up connections were activated. Map neuron M3 was now active because of the steering activity from the pointer neurons. In phase 4, the pointer neurons P1 and P2 were stimulated by attentional inputs of frequencies [700Hz,0Hz] which amplified the activity of M3 but the map activity returned back to the activity shown in phase 3 once the attentional inputs were withdrawn in phase 5. (b) Results from modified architecture. The sensory inputs to M3 and M7 were of frequencies [200Hz,230Hz] for the red curve and [40Hz,50Hz] for the blue curve. The 5 phases in the experiment were as described in (a). However in phase 5, we could see that map neuron M3 retained its activity even after the attentional inputs were withdrawn (attentional inputs to P1 and P2 were [700Hz,0Hz] for the red curve and [300Hz,0Hz] for the blue curve).

## Acknowledgments

The authors would like to thank M. Oster for help with setting up the AER infrastructure, S. Zahnd for the PCB design, and T. Delbrück for discussions on the digital-to-analog converter circuits. This work is partially supported by ETH Research Grant TH-20/04-2, and EU grant "DAISY" FP6-2005-015803.

## References

[1] Y. X. Wang and S. C. Liu, "Programmable synaptic weights for an aVLSI network of spiking neurons," in *Proceedings of the 2006 IEEE International Symposium on Circuits and Systems*, pp. 4531–4534, 2006.

[2] R. Hahnloser, R. J. Douglas, M. A. Mahowald, and K. Hepp, "Feedback interactions between neuronal pointers and maps for attentional processing," *Nature Neuroscience*, vol. 2, pp. 746–752, 1999.

[3] K. A. Boahen, "Point-to-point connectivity between neuromorphic chips using address event," *IEEE Transactions on Circuits and Systems II*, vol. 47, pp. 416-434, 2000.

[4] S.-C. Liu, and R. Douglas, "Temporal coding in a network of silicon integrate-and-fire neuron," *IEEE Transactions on Neural Networks: Special Issue on Temporal Coding for Neural Information Processing*, vol 15, no 5, Sep., pp. 1305-1314, 2004.

[5] S. R. Deiss, R. J. Douglas, and A. M. Whatley, "A pulse-coded communications infrastructure for neuromorphic systems," in *Pulsed Neural Networks*, W. Maass and C. M. Bishop, Eds. Boston, MA: MIT Press, 1999, ch. 6, pp. 157–178, ISBN 0-262-13350-4.

[6] C. Itti, E. Niebur, and C. Koch, "A model of saliency-based fast visual attention for rapid scene analysis," *IEEE Transactions on Pattern Analysis and Machine Intelligence*, vol. 20, no. 11, pp. 1254–1259, Apr 1998.

[7] G. Indiveri, T. Horiuchi, E. Niebur, and R. Douglas, "A competitive network of spiking VLSI neurons," in *World Congress on Neuroinformatics*, F. Rattay, Ed.    Vienna, Austria: ARGESIM/ASIM Verlag, Sept 24–29 2001, aRGESIM Reports.

[8] M. Oster and S.-C. Liu, "Spiking inputs to a winner-take-all network," in *Advances in Neural Information Processing Systems*.   Cambridge, MA: MIT Press, 2006, vol. 18.
